# Directional Hearing by the Mauthner System

**Audrey L. Gusik**
Department of Psychology
University of Colorado
Boulder, Co. 80309

**Robert C. Eaton**
E. P. O. Biology
University of Colorado
Boulder, Co. 80309

## Abstract

We provide a computational description of the function of the Mauthner system. This is the brainstem circuit which initiates fast-start escapes in teleost fish in response to sounds. Our simulations, using backpropagation in a realistically constrained feedforward network, have generated hypotheses which are directly interpretable in terms of the activity of the auditory nerve fibers, the principle cells of the system and their associated inhibitory neurons.

## 1 INTRODUCTION

### 1.1 THE MAUTHNER SYSTEM

Much is known about the brainstem system that controls fast-start escapes in teleost fish. The most prominent feature of this network is the pair of large Mauthner cells whose axons cross the midline and descend down the spinal cord to synapse on primary motoneurons. The Mauthner system also includes inhibitory neurons, the PHP cells, which have a unique and intense field effect inhibition at the spike-initiating zone of the Mauthner cells (Faber and Korn, 1978). The Mauthner system is part of the full brainstem escape network which also includes two pairs of cells homologous to the Mauthner cell and other populations of reticulospinal neurons. With this network fish initiate escapes only from appropriate stimuli, turn away from the offending stimulus, and do so very rapidly with a latency around 15 msec in goldfish. The Mauthner cells play an important role in these functions. Only one fires thus controlling the direction of the initial turn, and it fires very quickly (4-5 msec). They also have high thresholds due to instrinsic membrane properties and the inhibitory influence of the PHP cells. (For reviews, see Eaton, et al, 1991 and Faber and Korn, 1978.)

Acoustic stimuli are thought to be sufficient to trigger the response (Blaxter, 1981), both Mauthner cells and PHP cells receive innervation from primary auditory fibers (Faber and Korn, 1978). In addition, the Mauthner cells have been shown physio­logically to be very sensitive to acoustic pressure (Canfield and Eaton, 1990).

## 1.2   LOCALIZING SOUNDS UNDERWATER

In contrast to terrestrial vertebrates, there are several reasons for supposing that fish do not use time of arrival or intensity differences between the two ears to localize sounds: underwater sound travels over four times as fast as in air; the fish body provides no acoustic shadow; and fish use a single transducer to sense pressure which is conveyed equally to the two ears. Sound pressure is transduced into vibrations by the swim bladder which, in goldfish, is mechanically linked to the inner ear.

Fish are sensitive to an additional component of the acoustic wave, the particle motion. Any particle of the medium taking part in the propagation of a longitudenal wave will oscillate about an equilibrium point along the axis of propagation. Fish have roughly the same density as water, and will experience these oscillations. The motion is detected by the bending of sensory hairs on auditory receptor cells by the otolith, an inertial mass suspended above the hair cells. This component of the sound will provide the axis of propagation, but there is a 180 degree ambiguity.

Both pressure and particle motion are sensed by hair cells of the inner ear. In goldfish these signals may be nearly segregated. The linkage with the swim bladder impinges primarily on a boney chamber containing two of the endorgans of the inner ear: the saccule and the lagena. The utricle is a third endorgan also thought to mediate some acoustic function, without such direct input from the swimbladder.

Using both of these components fish can localize sounds. According to the phase model (Schuijf, 1981) fish analyze the phase difference between the pressure com­ponent of the sound and the particle displacement component to calculate distance and direction. When pressure is increasing, particles will be pushed in the direc­tion of sound propagation , and when pressure is decreasing particles will be pulled back. There will be a phase lag between pressure and particle motion which varies with frequency and distance from the sound source. This, and the separation of the pressure from the displacement signals in the ear of some species pose the greatest problems for theories of sound localization in fish.

The acoustically triggered escape in goldfish is a uniquely tractable problem in underwater sound localization. First, there is the fairly good segregation of pressure from particle motion at the sensory level. Second, the escape is very rapid. The decision to turn left or right is equivalent to the firing of one or the other Mauthner cell, and this happens within about 4 msec. With transmission delay, this decision relies only on the initial 2 msec or so of the stimulus. For most salient frequencies, the phase lag will not introduce uncertainty: both the first and second derivatives of particle position and acoustic pressure will be either positive or negative.

## 1.3   THE XNOR MODEL

A

| Active pressure input | Active displacement input | Left Mauthner output | Right Mauthner output |
|---|---|---|---|
| P+ | DL | On | Off |
| P+ | DR | Off | On |
| P− | DL | Off | On |
| P− | DR | On | Off |

B

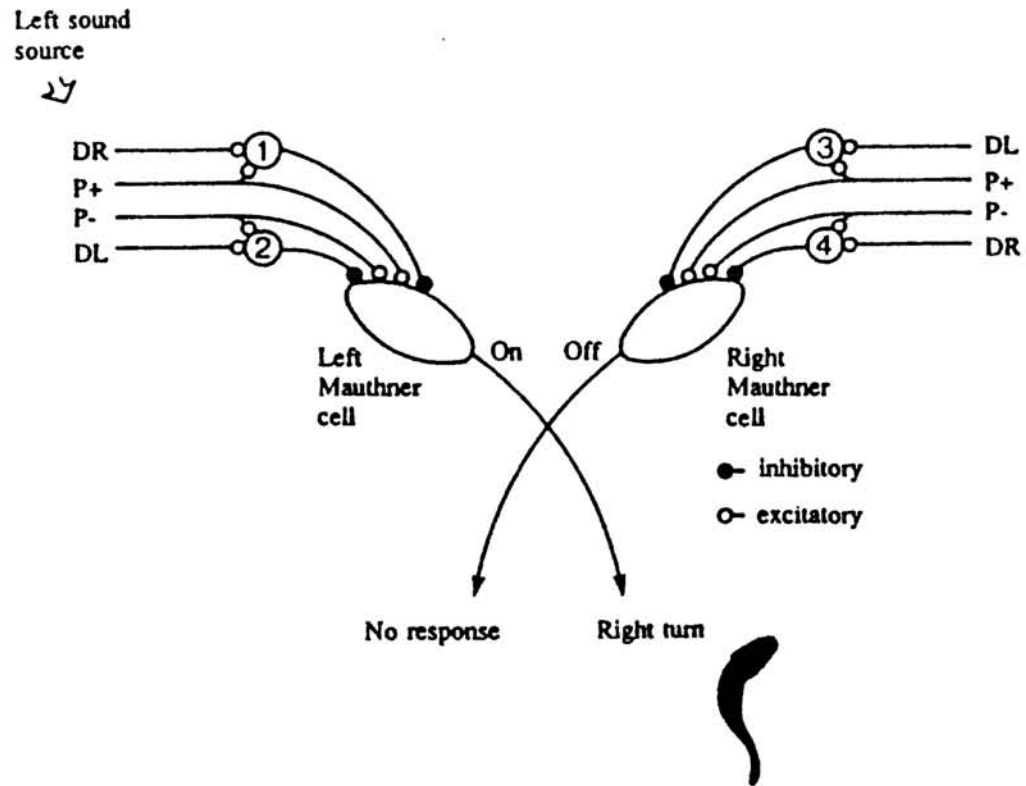

Figure 1 Truth table and minimal network for the XNOR model.

Given the above simplification of the problem, we can see that each Mauthner cell must perform a logical operation (Guzik and Eaton, 1993; Eaton et al, 1994). The left Mauthner cell should fire when sounds are located on the left, and this occurs when either pressure is increasing and particle motion is from the left or when pressure is decreasing and particle motion is from the right. We can call displacement from the left positive for the left Mauthner cell, and immediately we

have the logical operator exclusive-nor (or XNOR). The right Mauthner cell must solve the same problem with a redefinition of right displacement as positive. The conditions for this logic gate are shown in figure 1A for both Mauthner cells. This analysis simplifies our task of understanding the computational role of individual elements in the system. For example, a minimal network could appear as in figure 1B.

In this model PHP units perform a logical sub-task of the XNOR as AND gates. This model requires at least two functional classes of PHP units on each side of the brain. These PHP units will be activated for the combinations of pressure and displacement that indicate a sound coming from the wrong direction for the Mauthner cell on that side. Both Mauthner cells are activated by sufficient changes in pressure in either direction, high or low, and will be gated by the PHP cells. This minimal model emerged from explorations of the system using the connectionist paradigm, and inspired us to extend our efforts to a more realistic context.

## 2   THE NETWORK

We used a connectionist model to explore candidate solutions to the left/right discrimination problem that include the populations of neurons known to exist and include a distributed input resembling the sort available from the hair cells of the inner ear. We were interested in generating a number of alternative solutions to be better prepared to interpret physiological recordings from live goldfish, and to look for variations of, or alternatives to, the XNOR model.

### 2.1   THE ARCHITECTURE

As shown in figure 2, there are four layers in the connectionist model. The input layer consists of four pools of hair cell units. These represent the sensory neurons of the inner ear. There are two pools on each side: the saccule and the utricle. Treating only the horizontal plane, we have ignored the lagena in this model. The saccule is the organ of pressure sensation and the utricle is treated as the organ of particle motion. Each pool contains 16 hair cell units maximally responsive for displacements of their sensory hairs in one particular direction. They are activated as the cosine of the difference between their preferred direction and the stimulus deflection. All other units use sigmoidal activation functions.

The next layer consists of units representing the auditory fibers of the VIIIth nerve. Each pool receives inputs from only one pool of hair cell units, as nerve fibers have not been seen to innervate more than one endorgan. There are 10 units per fiber pool.

The fiber units provide input to both the inhibitory PHP units, and to the Mauthner units. There are four pools of PHP units, two on each side of the fish. One set on each side represents the collateral PHP cells, and the other set represents the commissural PHP cells (Faber and Korn, 1978). Both types receive inputs from the auditory fibers. The collaterals project only to the Mauthner cell on the same side. The commissurals project to both Mauthner cells. There are five units per PHP pool.

The Mauthner cell units receive inputs from saccular and utricular fibers on their same side only, as well as inputs from a single collateral PHP population and both commissural PHP populations.

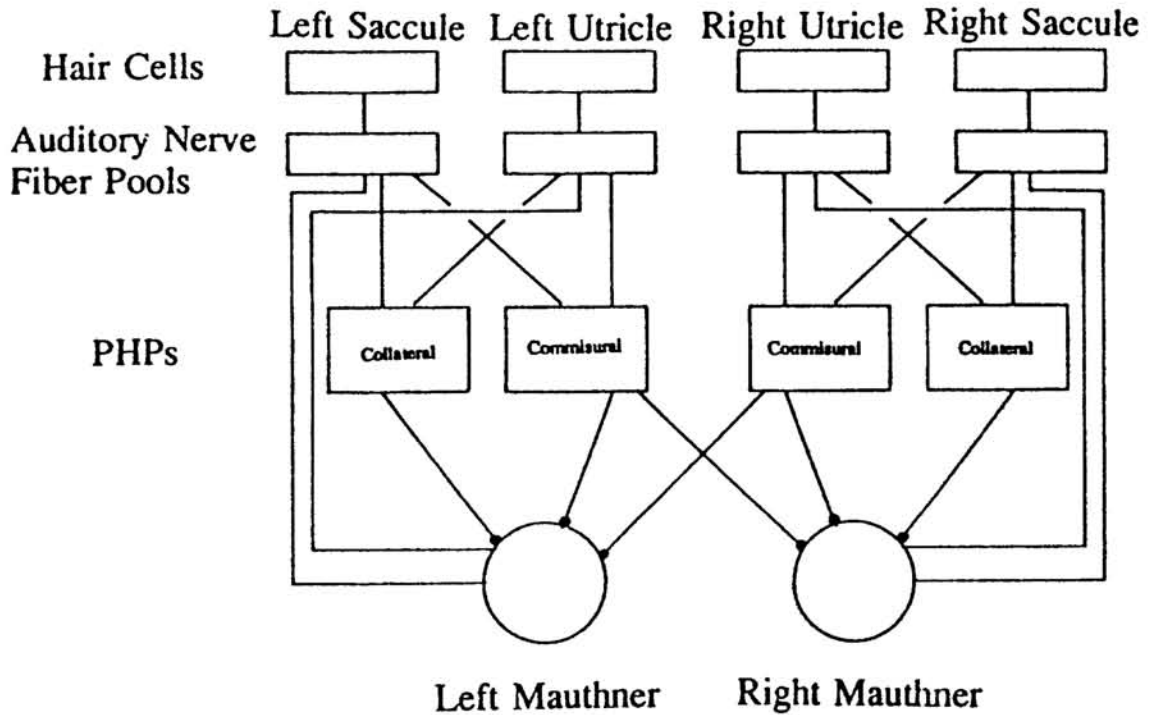

Figure 2 The architecture.

Weights from the PHP units are all constrained to be negative, while all others are constrained to be positive. The weights are implemented using the function below, positive or negative depending on the polarity of the weight.

$$f(w) = 1/2 \ (w + \ln \cosh(w) + \ln 2)$$

The function asymptotes to zero for negative values, and to the identity function for values above 2. This function vastly improved learning compared with the simpler, but highly nonlinear exponential function used in earlier versions of the model.

## 2.2  TRAINING

We used a total of 240 training examples. We began with a set of 24 directions for particle motion, evenly distributed around 360 degrees. These each appeared twice, once with increasing pressure and once with decreasing pressure, making a base set of 48 examples. Pressure was introduced as a deflection across saccular hair cells of either 0 degrees for low pressure, or 180 degrees for high pressure. These should be thought of as reflecting the expansion or compression of the swim bladder. Targets for the Mauthner cells were either 0 or 1 depending upon the conditions as described in the XNOR model, in figure 1A.

Next by randomly perturbing the activations of the hair cells for these 48 patterns, we generated 144 noisy examples. These were randomly increased or decreased up to 10%. An additional 48 examples were generated by dividing the hair cell activity by two to represent sub-threshold stimuli. These last 48 targets were set to zero.

The network was trained in batch mode with backpropagation to minimize a cross-entropy error measure, using conjugate gradient search. Unassisted backpropagation was unsuccessful at finding solutions.

For the eight solutions discussed here, two parameters were varied at the inputs. In some solutions the utricle was stimulated with a vector sum of the displacement and the pressure components, or a "mixed" input. In some solutions the hair cells in the utricle are not distributed uniformly, but in a gaussian manner with the mean tuning of 45 degrees to the right or left, in the two ears respectively. This approximates the actual distribution of hair cells in the goldfish utricle (Platt, 1977).

## 3   RESULTS

Analyzing the activation of the hidden units as a function of input pattern we found activity consistent with known physiology, nothing inconsistent with our knowledge of the system, and some predictions to be evaluated during intracellular recordings from PHP cells and auditory afferents.

First, many PHP cells were found exhibiting a logical function, which is consistent with our minimal model described above. These tended to project only to one Mauthner cell unit, which suggests that primarily the collateral PHP cells will demonstrate logical properties. Most logical PHP units were NAND gates with very large weights to one Mauthner cell. An example is a unit which is on for all stimuli except those having displacements anywhere on the left when pressure is high.

Second, saccular fibers tended to be either sensitive to high or low pressure, consistent with recordings of Furukawa and Ishii (1967). In addition there were a class which looked like threshold fibers, highly active for all supra-threshold stimuli, and inactive for all sub-threshold stimuli. There were some fibers with no obvious selectivity, as well.

Third, utricular fibers often demonstrate sensitivity for displacements exclusively from one side of the fish, consistent with our minimal model. Right and left utricular fibers have not yet been demonstrated in the real system.

Utricular fibers also demonstrated more coarsely tuned, less interpretable receptive fields. All solutions that included a mixed input to the utricle, for example, produced fibers that seemed to be "not 180 degree",or "not 0 degree", countering the pressure vectors. We interpret these fibers as doing clean-up given the absence of negative weights at that layer.

Fourth, sub-threshold behavior of units is not always consistent with their supra-threshold behavior. At sub-threshold levels of stimulation the activity of units may not reflect their computational role in the behavior. Thus, intracellular recordings should explore stimulus ranges known to elicit the behavior.

Fifth, Mauthner units usually receive very strong inputs from pressure fibers. This is consistent with physiological recordings which suggest that the Mauthner cells in goldfish are more sensitive to sound pressure than displacement (Canfield and Eaton, 1990).

Sixth, Mauthner cells always acquired relatively equal high negative biases. This is consistent with the known low input resistance of the real Mauthner cells, giving them a high threshold (Faber and Korn, 1978).

Seventh, PHP cells that maintain substantial bilateral connections tend to be tonically active. These contribute additional negative bias to the Mauthner cells. The relative sizes of the connections are often assymetric. This suggests that the commissural PHP cells serve primarily to regulate Mauthner threshold, ensure behavioral response only to intense stimuli, consistent with Faber and Korn (1978). These cells could only contribute to a partial solution of the XNOR problem.

Eighth, all solutions consistently used logic gate PHP units for only 50% to 75% of the training examples. Probably distributed solutions relying on the direct connections of auditory nerve fibers to Mauthner cells were more easily learned, and logic gate units only developed to handle the unsolved cases. Cases solved without logic gate units were solved by assymetric projections to the Mauthner cells of one polarity of pressure and one class of direction fibers, left or right.

Curiously, most of these cases involved a preferential projection from high pressure fibers to the Mauthner units, along with directional fibers encoding displacements from each Mauthner unit's positive direction. This means the logic gate units tended to handle the low pressure cases. This may be a result of the presence of the assymetric distributions of utricular hair cells in 6 out of the 8 solutions.

## 4   CONCLUSIONS

We have generated predictions for the behavior of neurons in the Mauthner system under different conditions of acoustic stimulation. The predictions generated with our connectionist model are consistent with our interpretation of the phase model for underwater sound localization in fishes as a logical operator. The results are also consistent with previously described properties of the Mauthner system. Though perhaps based on the characteristics more of the training procedure, our solutions suggest that we may find a mixed solution in the fish. Direct projections to the Mauthner cells from the auditory nerve perhaps handle many of the commonly encountered acoustic threats. The results of Blaxter (1981) support the idea that fish do escape from stimuli regardless of the polarity of the initial pressure change. Without significant nonlinear processing at the Mauthner cell itself, or more complex processing in the auditory fibers, direct connections could not handle all of these cases. These possibilities deserve exploration.

We propose different computational roles for the two classes of inhibitory PHP neurons. We expect only unilaterally-projecting PHP cells to demonstrate some logical function of pressure and particle motion. We believe that some elements of the Mauthner system must be found to demonstrate such minimal logical functions if the phase model is an explanation for left-right discrimination by the Mauthner system.

We are currently preparing to deliver controlled acoustic stimuli to goldfish during acute intracellular recording procedures from the PHP neurons, the afferent fibers and the Mauthner cells. Our insights from this model will greatly assist us in designing the stimulus regimen, and in interpreting our experimental results. Plans for future computational work are of a dynamic model that will include the results of these physiological investigations, as well as a more realistic version of the Mauthner cell.

## Acknowledgements

We are grateful for the technical assistance of members of the Boulder Connectionist Research Group, especially Don Mathis for help in debugging and optimizing the original code. We thank P.L. Edds-Walton for crucial discussions. This work was supported by a grant to RCE from the National Institutes of Health (RO1 NS22621).

## References

Blaxter, J.H.S., J.A.B. Gray, and E.J. Denton (1981) Sound and startle responses in herring shoals. J. Mar. Biol. Assoc. UK, *61*: 851-869

Canfield, J.G. and R.C. Eaton (1990) Swimbladder acoustic pressure transduction intiates Mauthner-mediated escape. Nature, *347*: 760-762

Eaton, R.C., J.G. Canfield and A.L. Guzik (1994) Left-right discrimination of sound onset by the Mauthner system. Brain Behav. Evol., *in press*

Eaton, R.C., R. DiDomenico and J. Nissanov (1991) Role of the Mauthner cell in sensorimotor integration by the brain stem escape network. Brain Behav. Evol., *37*: 272-285

Faber, D.S. and H. Korn (1978) Electrophysiology of the Mauthner cell: Basic properties, synaptic mechanisms and associated networks. *In* Neurobiology of the Mauthner Cell, D.S. Faber and H. Korn (eds), Raven Press, NY, pp. 47-131

Fay, R.R.(1984) The goldfish ear codes the axis of acoustic particle motion in three dimensions. Science, *225*: 951-954

Furukawa, T. and Y. Ishii (1967) Effects of static bending of sensory hairs on sound reception in the goldfish. Japanese J. Physiol., *17*: 572-588

Guzik, A.L. and R.C. Eaton (1993) The XNOR model for directional hearing by the Mauthner system. Soc. Neurosci. Abstr.

Platt, C. (1977) Hair cell distribution and orientation in goldfish otolith organs. J. Comp. Neurol., *172*: 283-298

Schuijf, A. (1981) Models of acoustic localization. *In* Hearing and Sound Communication in Fishes, W.N. Tavolga, A.N. Popper and R.R. Fay (eds.), Springer, New York,. pp. 267-310